# Asymptotics of Gaussian Regularized Least-Squares

**Ross A. Lippert**
M.I.T., Department of Mathematics
77 Massachusetts Avenue
Cambridge, MA 02139-4307
lippert@math.mit.edu

**Ryan M. Rifkin**
Honda Research Institute USA, Inc.
145 Tremont Street
Boston, MA 02111
rrifkin@honda-ri.com

## Abstract

We consider regularized least-squares (RLS) with a Gaussian kernel. We prove that if we let the Gaussian bandwidth $\sigma \to \infty$ while letting the regularization parameter $\lambda \to 0$, the RLS solution tends to a polynomial whose order is controlled by the rielative rates of decay of $\frac{1}{\sigma^2}$ and $\lambda$: if $\lambda = \sigma^{-(2k+1)}$, then, as $\sigma \to \infty$, the RLS solution tends to the $k$th order polynomial with minimal empirical error. We illustrate the result with an example.

## 1   Introduction

Given a data set $(x_1, y_1), (x_2, y_2), \ldots, (x_n, y_n)$, the inductive learning task is to build a function $f(x)$ that, given a new $x$ point, can predict the associated $y$ value. We study the Regularized Least-Squares (RLS) algorithm for finding $f$, a common and popular algorithm [2, 5] that can be used for either regression or classification:

$$\min_{f \in \mathcal{H}} \frac{1}{n} \sum_{i=1}^{n} (f(x_i) - y_i)^2 + \lambda ||f||_K^2.$$

Here, $\mathcal{H}$ is a Reproducing Kernel Hilbert Space (RKHS) [1] with associated kernel function $K$, $||f||_K^2$ is the squared norm in the RKHS, and $\lambda$ is a regularization constant controlling the tradeoff between fitting the training set accurately and forcing smoothness of $f$.

The Representer Theorem [7] proves that the RLS solution will have the form $f(x) = \sum_{i=1}^{n} c_i K(x_i, x)$, and it is easy to show [5] that we can find the coefficients $c$ by solving the linear system

$$(K + \lambda n I)c = y, \tag{1}$$

where $K$ is the $n$ by $n$ matrix satisfying $K_{ij} = K(x_i, x_j)$. We focus on the Gaussian kernel $K(x_i, x_j) = \exp(-||x_i - x_j||^2 / 2\sigma^2)$.

Our work was originally motivated by the empirical observation that on a range of benchmark classification tasks, we achieved surprisingly accurate classification using a Gaussian kernel with a very large $\sigma$ and a very small $\lambda$ (Figure 1; additional examples in [6]). This prompted us to study the large-$\sigma$ asymptotics of RLS. As $\sigma \to \infty$, $K(x_i, x_j) \to 1$ for arbitrary $x_i$ and $x_j$. Consider a single test point $x_0$. RLS will first find $c$ using Equation 1,

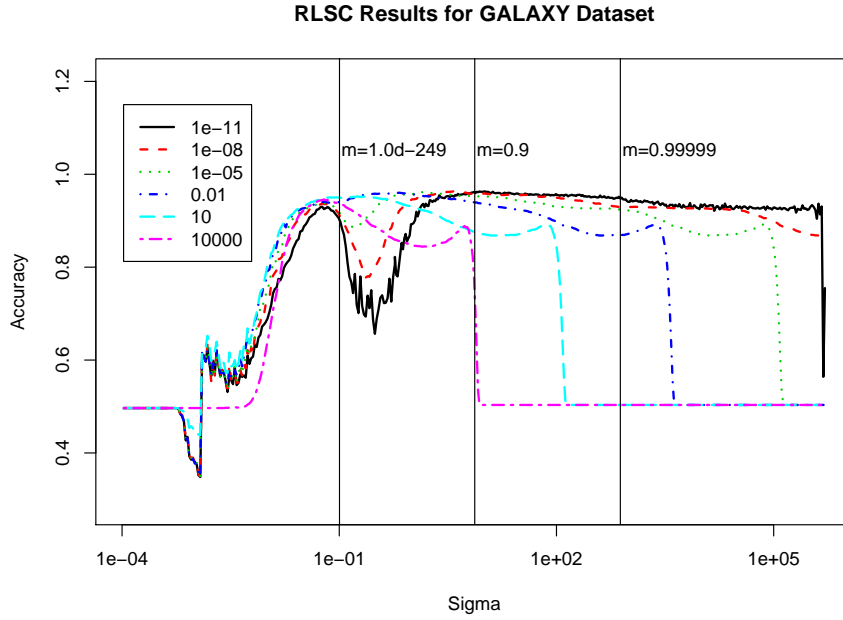

**Fig. 1.** RLS classification accuracy results for the UCI Galaxy dataset over a range of $\sigma$ (along the $x$-axis) and $\lambda$ (different lines) values. The vertical labelled lines show $m$, the smallest entry in the kernel matrix for a given $\sigma$. We see that when $\lambda = 1e - 11$, we can classify quite accurately when the smallest entry of the kernel matrix is .99999.

then compute $f(x_0) = c^t k$ where $k$ is the kernel vector, $k_i = K(x_i, x_0)$. Combining the training and testing steps, we see that $f(x_0) = y^t (K + \lambda n I)^{-1} k$.

Both $K$ and $k$ are close to 1 for large $\sigma$, i.e. $K_{ij} = 1 + \epsilon_{ij}$ and $k_i = 1 + \epsilon_i$. If we directly compute $c = (K + \lambda n I)^{-1} y$, we will tend to wash out the effects of the $\epsilon_{ij}$ term as $\sigma$ becomes large. If, instead, we compute $f(x_0)$ by associating to the right, first computing *point affinities* $(K + \lambda n I)^{-1} k$, then the $\epsilon_{ij}$ and $\epsilon_j$ interact meaningfully; this interaction is crucial to our analysis.

Our approach is to Taylor expand the kernel elements (and thus $K$ and $k$) in $1/\sigma$, noting that as $\sigma \to \infty$, consecutive terms in the expansion differ enormously. In computing $(K + \lambda n I)^{-1} k$, these scalings cancel each other out, and result in finite point affinities even as $\sigma \to \infty$. The asymptotic affinity formula can then be "transposed" to create an alternate expression for $f(x_0)$. Our main result is that if we set $\sigma^2 = s^2$ and $\lambda = s^{-(2k+1)}$, then, as $s \to \infty$, the RLS solution tends to the $k$th order polynomial with minimal empirical error.

The main theorem is proved in full. Due to space restrictions, the proofs of supporting lemmas and corollaries are omitted; an expanded version containing all proofs is available [4].

## 2 Notation and definitions

**Definition 1.** *Let $x_i$ be a set of $n + 1$ points ($0 \leq i \leq n$) in a $d$ dimensional space. The scalar $x_{ia}$ denotes the value of the $a^{th}$ vector component of the $i^{th}$ point.*

*The $n \times d$ matrix, $X$ is given by $X_{ia} = x_{ia}$.*

We think of $X$ as the matrix of training data $x_1, \ldots, x_n$ and $x_0$ as an $1 \times d$ matrix consisting of the test point.

Let $1_m, 1_{lm}$ denote the $m$ dimensional vector and $l \times m$ matrix with components all 1, similarly for $0_m, 0_{lm}$. We will dispense with such subscripts when the dimensions are clear from context.

**Definition 2 (Hadamard products and powers).** *For two $l \times m$ matrices, $N, M$, $N \odot M$ denotes the $l \times m$ matrix given by $(N \odot M)_{ij} = N_{ij} M_{ij}$. Analogously, we set $(N^{\odot c})_{ij} = N_{ij}^c$.*

**Definition 3 (polynomials in the data).** *Let $I \in \mathbb{Z}_{\geq 0}^d$ (non-negative multi-indices) and $Y$ be a $k \times d$ matrix. $Y^I$ is the $k$ dimensional vector given by $\left(Y^I\right)_i = \prod_{a=1}^d Y_{ia}^{I_a}$. If $h : \mathbb{R}^d \to \mathbb{R}$ then $h(Y)$ is the $k$ dimensional vector given by $(h(Y))_i = h(Y_{i1}, \ldots, Y_{id})$.*

*The $d$ canonical vectors, $e_a \in \mathbb{Z}_{\geq 0}^d$, are given by $(e_a)_b = \delta_{ab}$.*

Any scalar function, $f : \mathbb{R} \to \mathbb{R}$, applied to any matrix or vector, $A$, will be assumed to denote the elementwise application of $f$. We will treat $y \to e^y$ as a scalar function (we have no need of matrix exponentials in this work, so the notation is unambiguous).

We can re-express the kernel matrix and kernel vector in this notation:

$$K = e^{\frac{1}{2\sigma^2} \sum_{a=1}^d 2X^{e_a}(X^{e_a})^t - X^{2e_a}1_n^t - 1_n\left(X^{2e_a}\right)^t} \tag{2}$$

$$= \mathrm{diag}\left(e^{-\frac{1}{2\sigma^2}||X||^2}\right) e^{\frac{1}{\sigma^2}XX^t} \mathrm{diag}\left(e^{-\frac{1}{2\sigma^2}||X||^2}\right) \tag{3}$$

$$k = e^{\frac{1}{2\sigma^2} \sum_{a=1}^d 2X^{e_a}x_0^{e_a} - X^{2e_a}1_1 - 1_n x_0^{2e_a}} \tag{4}$$

$$= \mathrm{diag}\left(e^{-\frac{1}{2\sigma^2}||X||^2}\right) e^{\frac{1}{\sigma^2}Xx_0^t} e^{-\frac{1}{2\sigma^2}||x_0||^2}. \tag{5}$$

## 3 Orthogonal polynomial bases

Let $V_c = \mathrm{span}\{X^I : |I| = c\}$ and $V_{\leq c} = \bigcup_{a=0}^c V_c$ which can be thought of as the set of all $d$ variable polynomials of degree $c$, evaluated on the training data. Since the data are finite, there exists $b$ such that $V_{\leq c} = V_{\leq b}$ for all $c \geq b$. Generically, $b$ is the smallest $c$ such that $\binom{c+d}{d} \geq n$.

Let $Q$ be an orthonormal matrix in $\mathbb{R}^{n \times n}$ whose columns progressively span the $V_{\leq c}$ spaces, i.e. $Q = (\begin{array}{cccc} B_0 & B_1 & \cdots & B_b \end{array})$ where $Q^t Q = I$ and $\mathrm{colspan}\{(\begin{array}{ccc} B_0 & \cdots & B_c \end{array})\} = V_{\leq c}$. We might imagine building such a $Q$ via the Gramm-Schmidt process on the vectors $X^0, X^{e_1}, \ldots, X^{e_d}, \ldots X^I, \ldots$ taken in order of non-decreasing $|I|$.

Letting $C_I = \binom{|I|}{I_1 \ldots I_d}$ be multinomial coefficients, the following relations between $Q, X$, and $x_0$ are easily proved.

$$(Xx_0^t)^{\odot c} = \sum_{|I|=c} C_I X^I (x_0^I)^t \quad \text{hence} \quad (Xx_0^t)^{\odot c} \in V_c$$

$$(XX^t)^{\odot c} = \sum_{|I|=c} C_I X^I (X^I)^t \quad \text{hence} \quad \mathrm{colspan}\{(XX^t)^{\odot c}\} = V_c$$

and thus, $B_i^t(Xx_0^t)^{\odot c} = 0$ if $i > c$, $B_i^t(XX^t)^{\odot c}B_j = 0$ if $i > c$ or $j > c$, and $B_c^t(XX^t)^{\odot c}B_c$ is non-singular.

Finally, we note that $\text{argmin}_{v \in V_{\leq c}}\{||y - v||\} = \sum_{a \leq c} B_a(B_a^t y)$.

## 4 Taking the $\sigma \to \infty$ limit

We will begin with a few simple lemmas about the limiting solutions of linear systems. At the end of this section we will arrive at the limiting form of suitably modified RLSC equations.

**Lemma 1.** *Let $i_1 < \cdots < i_q$ be positive integers. Let $A(s), y(s)$ be a block matrix and block vector given by*

$$A(s) = \begin{pmatrix} A_{00}(s) & s^{i_1}A_{01}(s) & \cdots & s^{i_q}A_{0q}(s) \\ s^{i_1}A_{10}(s) & s^{i_1}A_{11}(s) & \cdots & s^{i_q}A_{1q}(s) \\ \cdots & \cdots & \cdots & \cdots \\ s^{i_q}A_{q0}(s) & s^{i_q}A_{q1}(s) & \cdots & s^{i_q}A_{qq}(s) \end{pmatrix}, \quad y(s) = \begin{pmatrix} b_0(s) \\ s^{i_1}b_1(s) \\ \cdots \\ s^{i_q}b_q(s) \end{pmatrix}$$

*where $A_{ij}(s)$ and $b_i(s)$ are continuous matrix-valued and vector-valued functions of $s$ with $A_{ii}(0)$ non-singular for all $i$.*

$$\lim_{s \to 0} A^{-1}(s)y(s) = \begin{pmatrix} A_{00}(0) & 0 & \cdots & 0 \\ A_{10}(0) & A_{11}(0) & \cdots & 0 \\ \cdots & \cdots & \cdots & \cdots \\ A_{q0}(0) & A_{q1}(0) & \cdots & A_{qq}(0) \end{pmatrix}^{-1} \begin{pmatrix} b_0(0) \\ b_1(0) \\ \cdots \\ b_q(0) \end{pmatrix}$$

We are now ready to state and prove the main result of this section, characterizing the limiting large-$\sigma$ solution of Gaussian RLS.

**Theorem 1.** *Let $q$ be an integer satisfying $q < b$, and let $p = 2q + 1$. Let $\lambda = C\sigma^{-p}$ for some constant $C$. Define $A_{ij}^{(c)} = \frac{1}{c!}B_i^t(XX^t)^{\odot c}B_j$, and $b_i^{(c)} = \frac{1}{c!}B_i^t(Xx_0^t)^{\odot c}$.*

$$\lim_{\sigma \to \infty} \left(K + nC\sigma^{-p}I\right)^{-1}k = v$$

*where*

$$v = \begin{pmatrix} B_0 & \cdots & B_q \end{pmatrix}w \tag{6}$$

$$\begin{pmatrix} b_0^{(0)} \\ b_1^{(1)} \\ \cdots \\ b_q^{(q)} \end{pmatrix} = \begin{pmatrix} A_{00}^{(0)} & 0 & \cdots & 0 \\ A_{10}^{(1)} & A_{11}^{(1)} & \cdots & 0 \\ \cdots & \cdots & \cdots & \cdots \\ A_{q0}^{(q)} & A_{q1}^{(q)} & \cdots & A_{qq}^{(q)} \end{pmatrix}w \tag{7}$$

We first manipulate the equation $(K + n\lambda I)y = k$ according to the factorizations in (3) and (5).

$$K = \text{diag}\left(e^{-\frac{1}{2\sigma^2}||X||^2}\right)e^{\frac{1}{\sigma^2}XX^t}\text{diag}\left(e^{-\frac{1}{2\sigma^2}||X||^2}\right) = NPN$$

$$k = \text{diag}\left(e^{-\frac{1}{2\sigma^2}||X||^2}\right)e^{\frac{1}{\sigma^2}Xx_0^t}e^{-\frac{1}{2\sigma^2}||x_0||^2} = Nw\alpha$$

Noting that $\lim_{\sigma \to \infty} e^{-\frac{1}{2\sigma^2}||x_0||^2} \text{diag}\left(e^{\frac{1}{2\sigma^2}||X||^2}\right) = \lim_{\sigma \to \infty} \alpha N^{-1} = I$,
we have

$$
\begin{aligned}
v &\equiv \lim_{\sigma \to \infty} (K + nC\sigma^{-p}I)^{-1}k \\
&= \lim_{\sigma \to \infty} (NPN + \beta I)^{-1} Nw\alpha \\
&= \lim_{\sigma \to \infty} \alpha N^{-1}(P + \beta N^{-2})^{-1}w \\
&= \lim_{\sigma \to \infty} \left(e^{\frac{1}{\sigma^2}XX^t} + nC\sigma^{-p}\text{diag}\left(e^{\frac{1}{\sigma^2}||X||^2}\right)\right)^{-1} e^{\frac{1}{\sigma^2}Xx_0^t}.
\end{aligned}
$$

Changing bases with $Q$,

$$
Q^t v = \lim_{\sigma \to \infty} \left(Q^t e^{\frac{1}{\sigma^2}XX^t}Q + nC\sigma^{-p}Q^t\text{diag}\left(e^{\frac{1}{\sigma^2}||X||^2}\right)Q\right)^{-1} Q^t e^{\frac{1}{\sigma^2}Xx_0^t}.
$$

Expanding via Taylor series and writing in block form (in the $b \times b$ block structure of $Q$),

$$
Q^t e^{\frac{1}{\sigma^2}XX^t}Q = Q^t(XX^t)^{\odot 0}Q + \frac{1}{1!\sigma^2}Q^t(XX^t)^{\odot 1}Q + \frac{1}{2!\sigma^4}Q^t(XX^t)^{\odot 2}Q + \cdots
$$

$$
= \begin{pmatrix} A_{00}^{(0)} & 0 & \cdots & 0 \\ 0 & 0 & \cdots & 0 \\ \cdots & \cdots & \cdots & \cdots \\ 0 & 0 & \cdots & 0 \end{pmatrix} + \frac{1}{\sigma^2}\begin{pmatrix} A_{00}^{(1)} & A_{01}^{(1)} & \cdots & 0 \\ A_{10}^{(1)} & A_{11}^{(1)} & \cdots & 0 \\ \cdots & \cdots & \cdots & \cdots \\ 0 & 0 & \cdots & 0 \end{pmatrix} + \cdots
$$

$$
Q^t e^{\frac{1}{\sigma^2}Xx_0^t} = Q^t(Xx_0^t)^{\odot 0} + \frac{1}{\sigma^2}Q^t(Xx_0^t)^{\odot 1} + \frac{1}{\sigma^4}Q^t(Xx_0^t)^{\odot 2} + \cdots
$$

$$
= \begin{pmatrix} b_0^{(0)} \\ 0 \\ \cdots \\ 0 \end{pmatrix} + \frac{1}{\sigma^2}\begin{pmatrix} b_0^{(1)} \\ b_1^{(1)} \\ \cdots \\ 0 \end{pmatrix} + \cdots
$$

$$
nC\sigma^{-p}Q^t\text{diag}\left(e^{\frac{1}{\sigma^2}||X||^2}\right)Q = nC\sigma^{-p}I + \cdots.
$$

Since the $A_{cc}^{(c)}$ are non-singular, Lemma 3 applies, giving our result. $\square$

# 5 The classification function

When performing RLS, the actual prediction of the limiting classifier is given via

$$
f_\infty(x_0) \equiv \lim_{\sigma \to \infty} y^t(K + nC\sigma^{-p}I)^{-1}k.
$$

Theorem 1 determines $v = \lim_{\sigma \to \infty}(K + nC\sigma^{-p}I)^{-1}k$, showing that $f_\infty(x_0)$ is a polynomial in the training data $X$. In this section, we show that $f_\infty(x_0)$ is, in fact, a polynomial in the test point $x_0$. We continue to work with the orthonormal vectors $B_i$ as well as the auxilliary quantities $A_{ij}^{(c)}$ and $b_i^{(c)}$ from Theorem 1.

Theorem 1 shows that $v \in V_{\leq q}$: the point affinity function is a polynomial of degree $q$ in the training data, determined by (7).

$$
\sum_{i,j\leq c} c!B_i A_{ij}^{(c)} B_j^t = (XX^t)^{\odot c} \quad \text{hence} \quad \sum_{j\leq c} c!B_c A_{cj}^{(c)} B_j^t = B_c B_c^t(XX^t)^{\odot c}
$$

$$
\sum_{i\leq c} c!B_i b_i^{(c)} = (Xx_0^t)^{\odot c} \quad \text{hence} \quad c!B_c b_c^{(c)} = B_c B_c^t(Xx_0^t)^{\odot c}
$$

we can restate Equation 7 in an equivalent form:

$$
\begin{pmatrix} B_0^t \\ \cdots \\ B_q^t \end{pmatrix}^t \left( \begin{pmatrix} 0!b_0^{(0)} \\ 1!b_1^{(1)} \\ \cdots \\ q!b_q^{(q)} \end{pmatrix} - \begin{pmatrix} 0!A_{00}^{(0)} & 0 & \cdots & 0 \\ 1!A_{10}^{(1)} & 1!A_{11}^{(1)} & \cdots & 0 \\ \cdots & \cdots & \cdots & \cdots \\ q!A_{q0}^{(q)} & q!A_{q1}^{(q)} & \cdots & q!A_{qq}^{(q)} \end{pmatrix} \begin{pmatrix} B_0^t \\ \cdots \\ B_q^t \end{pmatrix} v \right) = 0 \quad (8)
$$

$$
\sum_{c \leq q} c! B_c b_c^{(c)} - \sum_{c \leq q} \sum_{j \leq c} c! B_c A_{cj}^{(c)} B_j^t v = 0 \quad (9)
$$

$$
\sum_{c \leq q} B_c B_c^t \left( (X x_0^t)^{\odot c} - (X X^t)^{\odot c} v \right) = 0. \quad (10)
$$

Up to this point, our results hold for arbitrary training data $X$. To proceed, we require a mild condition on our training set.

**Definition 4.** *$X$ is called* generic *if $X^{I_1}, \ldots, X^{I_n}$ are linearly independent for any distinct multi-indices $\{I_i\}$.*

**Lemma 2.** *For generic $X$, the solution to Equation 7 (or equivalently, Equation 10) is determined by the conditions $\forall I : |I| \leq q, (X^I)^t v = x_0^I$, where $v \in V_{\leq q}$.*

**Theorem 2.** *For generic data, let $v$ be the solution to Equation 10. For any $y \in \mathbb{R}^n$, $f(x_0) = y^t v = h(x_0)$, where $h(x) = \sum_{|I| \leq q} a_I x^I$ is a multivariate polynomial of degree $q$ minimizing $||y - h(X)||$.*

We see that as $\sigma \to \infty$, the RLS solution tends to the minimum empirical error $k$th order polynomial.

# 6 Experimental Verification

In this section, we present a simple experiment that illustrates our results. We consider a fith-degree polynomial function. Figure 2 plots $f$, along with a 150 point dataset drawn by choosing $x_i$ uniformly in $[0, 1]$, and choosing $y = f(x) + \epsilon_i$, where $\epsilon_i$ is a Gaussian random variable with mean 0 and standard deviation .05. Figure 2 also shows (in red) the best polynomial approximations to the data (not to the ideal $f$) of various orders. (We omit third order because it is nearly indistinguishable from second order.)

According to Theorem 1, if we parametrize our system by a variable $s$, and solve a Gaussian regularized least-squares problem with $\sigma^2 = s^2$ and $\lambda = C s^{-(2k+1)}$ for some integer $k$, then, as $s \to \infty$, we expect the solution to the system to tend to the $k$th-order data-based polynomial approximation to $f$. Asymptotically, the value of the constant $C$ does not matter, so we (arbitrarily) set it to be 1. Figure 3 demonstrates this result.

We note that these experiments frequently require setting $\lambda$ much smaller than machine-$\epsilon$. As a consequence, we need more precision than IEEE double-precision floating-point, and our results cannot be obtained via many standard tools (e.g., MATLAB(TM)) We performed our experiments using CLISP, an implementation of Common Lisp that includes arithmetic operations on arbitrary-precision floating point numbers.

# 7 Discussion

Our result provides insight into the asymptotic behavior of RLS, and (partially) explains Figure 1: in conjunction with additional experiments not reported here, we believe that

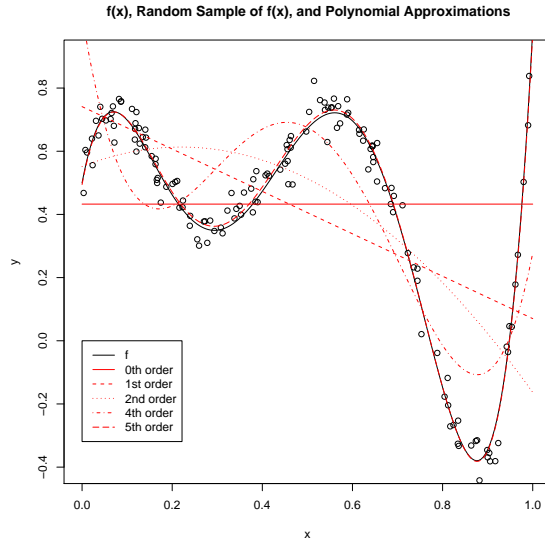

**f(x), Random Sample of f(x), and Polynomial Approximations**

**Fig. 2.** $f(x) = .5(1-x) + 150x(x - .25)(x - .3)(x - .75)(x - .95)$, a random dataset drawn from $f(x)$ with added Gaussian noise, and data-based polynomial approximations to $f$.

we are recovering second-order polynomial behavior, with the drop-off in performance at various $\lambda$'s occurring at the transition to third-order behavior, which cannot be accurately recovered in IEEE double-precision floating-point. Although we used the specific details of RLS in deriving our solution, we expect that in practice, a similar result would hold for Support Vector Machines, and perhaps for Tikhonov regularization with convex loss more generally.

An interesting implication of our theorem is that for very large $\sigma$, we can obtain various order polynomial classifications by sweeping $\lambda$. In [6], we present an algorithm for solving for a wide range of $\lambda$ for essentially the same cost as using a single $\lambda$. This algorithm is not currently practical for large $\sigma$, due to the need for extended-precision floating point.

Our work also has implications for approximations to the Gaussian kernel. Yang et al. use the Fast Gauss Transform (FGT) to speed up matrix-vector multiplications when performing RLS [8]. In [6], we studied this work; we found that while Yang et al. used moderate-to-small values of $\sigma$ (and did not tune $\lambda$), the FGT sacrificed substantial accuracy compared to the best achievable results on their datasets. We showed empirically that the FGT becomes much more accurate at larger values of $\sigma$; however, at large-$\sigma$, it seems likely we are merely recovering low-order polynomial behavior. We suggest that approximations to the Gaussian kernel must be checked carefully, to show that they produce sufficiently good results are moderate values of $\sigma$; this is a topic for future work.

## References

1. Aronszajn. Theory of reproducing kernels. *Transactions of the American Mathematical Society*, 68:337–404, 1950.

2. Evgeniou, Pontil, and Poggio. Regularization networks and support vector machines. *Advances In Computational Mathematics*, 13(1):1–50, 2000.

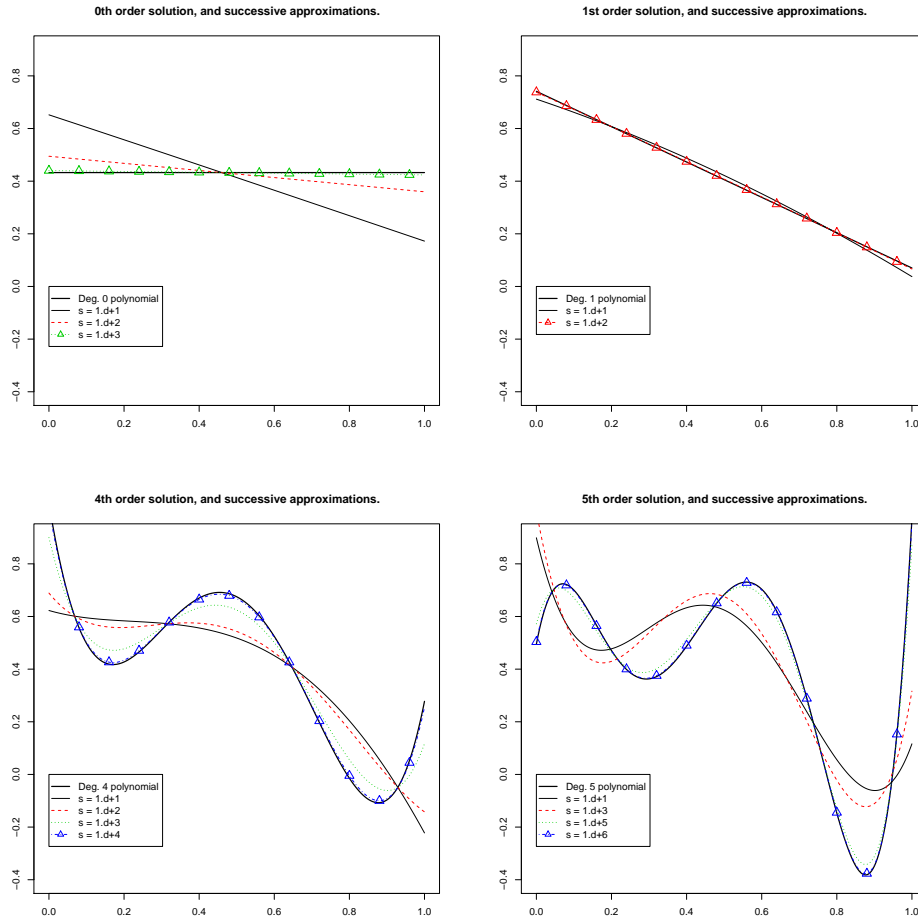

**Fig. 3.** As $s \to \infty$, $\sigma^2 = s^2$ and $\lambda = s^{-(2k+1)}$, the solution to Gaussian RLS approaches the $k$th order polynomial solution.

3. Keerthi and Lin. Asymptotic behaviors of support vector machines with gaussian kernel. *Neural Computation*, 15(7):1667–1689, 2003.

4. Ross Lippert and Ryan Rifkin. Asymptotics of gaussian regularized least-squares. Technical Report MIT-CSAIL-TR-2005-067, MIT Computer Science and Artificial Intelligence Laboratory, 2005.

5. Rifkin. *Everything Old Is New Again: A Fresh Look at Historical Approaches to Machine Learning*. PhD thesis, Massachusetts Institute of Technology, 2002.

6. Rifkin and Lippert. Practical regularized least-squares: $\lambda$-selection and fast leave-one-out-computation. In preparation, 2005.

7. Wahba. *Spline Models for Observational Data*, volume 59 of *CBMS-NSF Regional Conference Series in Applied Mathematics*. Society for Industrial & Applied Mathematics, 1990.

8. Yang, Duraiswami, and Davis. Efficient kernel machines using the improved fast Gauss transform. In *Advances in Neural Information Processing Systems*, volume 16, 2004.
